# StepbaQ: Stepping backward as Correction for Quantized Diffusion Models

**Yi-Chung Chen**[*]
MediaTek, Purdue University
chen5262@purdue.edu

**Zhi-Kai Huang**
MediaTek
brent5481@gmail.com

**Jing-Ren Chen**
MediaTek
jingren.chen@mediatek.com

## Abstract

Quantization of diffusion models has attracted considerable attention due to its potential to enable various applications on resource-constrained mobile devices. However, given the cumulative nature of quantization errors in quantized diffusion models, overall performance may still decline even with efforts to minimize quantization error at each sampling step. Recent studies have proposed several methods to address accumulated quantization error, yet these solutions often suffer from limited applicability due to their underlying assumptions or only partially resolve the issue due to an incomplete understanding. In this work, we introduce a novel perspective by conceptualizing quantization error as a "stepback" in the denoising process. We investigate how the accumulation of quantization error can distort the sampling trajectory, resulting in a notable decrease in model performance. To address this challenge, we introduce StepbaQ, a method that calibrates the sampling trajectory and counteracts the adverse effects of accumulated quantization error through a sampling step correction mechanism. Notably, StepbaQ relies solely on statistics of quantization error derived from a small calibration dataset, highlighting its strong applicability. Our experimental results demonstrate that StepbaQ can serve as a plug-and-play technique to enhance the performance of diffusion models quantized by off-the-shelf tools without modifying the quantization settings. For example, StepbaQ significantly improves the performance of the quantized SD v1.5 model by 7.30 in terms of FID on SDprompts dataset under the common W8A8 setting, and it enhances the performance of the quantized SDXL-Turbo model by 17.31 in terms of FID on SDprompts dataset under the challenging W4A8 setting.

## 1 Introduction

Diffusion models [11, 44, 23, 49] have demonstrated its power of generating high quality samples in a wide variety of applications including super resolution [53, 58], image enhancement [37, 54], image inpainting [56], image editing [28] and image-to-image translation [32]. In contrast to GAN [18] and VAE [30], which are prone to issues such as mode and posterior collapse, diffusion models consistently produce diverse, high-quality samples, thus emerging as the predominant technique in the field of image generation. However, the deployment of diffusion models on computationally constrained devices, such as smartphones, is impeded by their extensive computational demands, which stem from the complex network structures and the multitude of denoising steps required during the sampling phase. To enhance the computational efficiency of diffusion models, a body of research

---

[*]This work was conducted during Yi-Chung's time at MediaTek.

has concentrated on reducing the number of sampling steps [49, 39, 41, 46]. Concurrently, another strand of investigation has pursued model compression strategies, such as quantization [47, 33, 48, 19, 57, 24, 20, 52] and pruning [14], to diminish the computational resources necessitated. In this paper, we focus on improving the performance of model quantization.

Model quantization [31, 16] stands as one of the most popular model compression techniques. This technique transits model parameters and activations from a high bit-width floating-point format to a more compact low bit-width representation, thereby facilitating a substantial acceleration of model inference with a tolerable drop in performance. However, quantizing diffusion models leads to greater performance degradation compared to standard deep learning models due to two challenges: **i) Activation range varies across timesteps.** Diffusion models generate samples from a normal distribution through a multi-step denoising process, wherein noise levels are gradually reduced following a predefined schedule. This mechanism results in a significant disparity in the activation range across different timesteps, as noted in studies by Shang et al. [47] and Li et al. [33]. Such variability poses a challenge in establishing an appropriate quantization range. Opting for a broader interval may mitigate clipping errors at the expense of increased rounding errors and vice versa. Prior research by Shang et al. [47] and Li et al. [33] have proposed sampling strategies to select representative calibration samples over multiple timesteps. To further reduce quantization errors, So et al. [48], He et al. [19], and Yang et al. [57] relax the quantization constraints, adopting techniques such as timestep-specific quantization parameters and mixed precision quantization. However, since these methods do not consider the accumulation of quantization errors, they only achieve mediocre performance. **ii) Quantization error accumulates over time.** The iterative nature of the denoising process in diffusion models means that quantization errors from each step can accumulate. Although errors at individual sampling steps may appear negligible, the cumulative effect can significantly impair the final sample quality. To mitigate this error propagation, Huang et al. [24] have underscored the significance of temporal features and introduced a temporal information-aware reconstruction method to prevent deviations from the intended sampling trajectory caused by temporal information mismatch. Conversely, He et al. [20] have presented PTQD, which employs variance schedule calibration to integrate the quantization error at each step into the Gaussian diffusion noise of stochastic samplers, thereby addressing the accumulation issue. However, the approach proposed by Huang et al. [24] only resolves the disturbance of temporal features without considering the distribution shift in the latent space. On the other hand, the strategy of error absorption introduced by He et al. [20] is limited to stochastic samplers, thereby excluding its application to deterministic samplers, such as DDIM [49].

In this work, our goal is to devise a general approach capable of addressing the issue of error accumulation and enhancing the performance of the quantized diffusion model. We investigate the mechanisms by which accumulated errors can detrimentally impact overall model performance. Our research reveals that quantization error can alter the distribution of the latent variables. This shift in the distribution can lead to a divergence from the original sampling trajectory, ultimately causing a decline in model performance. Intriguingly, we show that with the assumption of quantization error following Gaussian distribution, the distribution shift in the latent space caused by quantization error can be interpreted as a "stepback" in the denoising process.

Building upon this insight, we introduce StepbaQ, a novel method that uses a sampling step correction technique to minimize the deviation from the original sampling trajectory. StepbaQ analyzes quantization errors to measure the extent of the distribution shifts and subsequently determine the "corrected sampling steps." We demonstrate that with appropriate modifications based on the corrected sampling steps, StepbaQ can effectively correct the sampling trajectory and mitigate the accumulation of quantization error, thereby significantly enhancing the quality of the generated results. It is worth noting that StepbaQ only necessitates the statistics of quantization error derived from a small calibration dataset. Consequently, it can be seamlessly integrated with existing quantization frameworks as a plug-and-play solution, enhancing the performance of diffusion models quantized by off-the-shelf tools without modifying their quantization settings.

The contributions of this paper are summarized as follows:

- We introduce a novel perspective that interprets the distribution shift in the latent space, caused by quantization error, as a "stepback" in the denoising process. We demonstrate that such a temporal shift can alter the sampling trajectory and adversely affect the generated results.

- We propose StepbaQ, a general strategy designed to enhance the performance of quantized diffusion models. This method employs a sampling step correction technique to realign the sampling trajectory and eliminate the accumulation of quantization error.

- Extensive experiments show that StepbaQ can serve as a plug-and-play technique, significantly improving the performance of diffusion models quantized by off-the-shelf tools, and achieve the state-of-the-art performance for quantized diffusion models.

## 2 Related Works

### 2.1 Efficient Diffusion Models

Although diffusion models have powerful generative capabilities, slow inference speeds severely limit their practical application. Various approaches have been developed to improve the efficiency of diffusion models. Rombach et al. [45] speeds up each denoising step by transferring the denoising process to the latent space; Song et al. [49] modifies the original diffusion process with a non-Markovian formulation, enabling fewer sampling steps and deterministic sampling. Lu et al. [39] further reduce the required number of steps by introducing an exact solution formulation for diffusion ODEs. Luo et al. [41] applies a consistency model to the latent space, enabling fast sampling with just a few steps or potentially a single step. Sauer et al. [46] leverages adversarial training on the diffusion model, enabling the network to generate images in one step, just like GAN [18]. While the methods above can effectively speed up the inference process of diffusion models, deploying these models on devices with limited computational resources necessitates reduced computation and memory usage, which can be achieved via model quantization.

### 2.2 Diffusion Model Quantization

Model quantization is a technique that effectively reduces memory and computation costs required for deep learning models. It can be divided into two categories: Quantization-Aware Training (QAT) [13, 17, 27, 26], which mitigates performance drop after quantization through model fine-tuning at the expense of substantial computational cost, and Post-Training Quantization (PTQ) [42, 34, 25, 55, 15, 43, 10], which requires only a small calibration dataset and therefore is ideal for large models like diffusion models. While prior works have achieved considerable progress, directly applying conventional quantization methods on diffusion models does not yield good results due to challenges such as varying activation range across steps and error accumulation. Shang et al. [47] and Li et al. [33] propose sampling strategies for selecting representative calibration samples across steps to alleviate the issue of varying activation ranges. Some methods relax the quantization constraints and use finer granularity to enhance quantization results. For example, Yang et al. [57] leverage SQNR as a metric to find sensitive blocks and quantize them with higher precision. Li et al. [35], So et al. [48], and He et al. [19] employ distinct sets of quantization parameters at various timesteps to address the drastic changes in activation ranges. While effective, these strategies incur additional computational overhead, which reduces their practicality. Moreover, even when efforts are made to minimize quantization error at each sampling step, the overall performance still suffers due to error accumulation. To address the issue of error accumulation, Huang et al. [24] propose temporal information aware reconstruction, which alleviates temporal feature disturbance, to prevent deviation from the original sampling trajectory. On the other hand, Tang et al. [50] introduces a progressive calibration strategy to minimize accumulation error. Aside from the approaches above, the method PTQD, proposed by He et al. [20], is closely related to our work. Both methods utilize the statistics of quantization error to enhance the performance of quantized diffusion models. They can be considered complementary techniques integrated with existing quantization approaches. PTQD decomposes the quantization error into correlated and uncorrelated parts. The correlated part is eliminated by simple division, while the uncorrelated part is absorbed into the scheduled Gaussian noise. Though this strategy is effective, its practicality is restricted. The design of the error absorption mechanism is only compatible with stochastic samplers, such as DDPM [23], which narrows its applicability. In contrast, our proposed StepbaQ, utilizing a sampling step correction technique, is applicable to both deterministic and stochastic samplers.

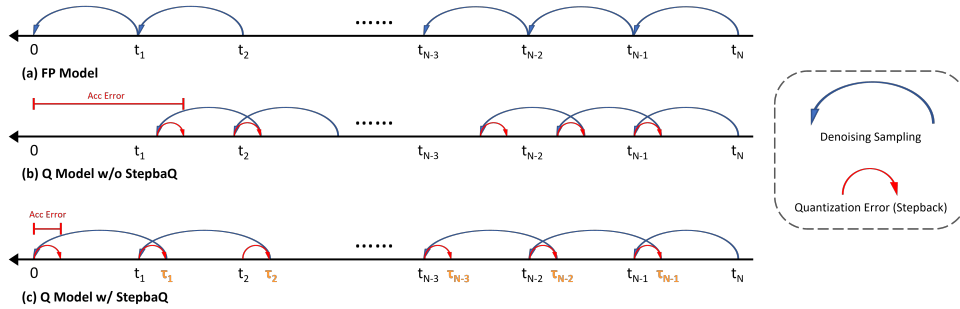

Figure 1: Overview of the denoising process of StepbaQ and existing methods. Figure (a) shows the original denoising process. Figure (b) demonstrates the negative impact of quantization error without changing the step size, leading to significant accumulation error. Figure (c), on the other hand, illustrates how StepbaQ treats the quantization error as a stepback in the denoising process and adopts corrected steps with a larger step size to eliminate cumulative quantization error.

## 3 Preliminary

Diffusion models [23, 49] are latent variable models characterized by a forward process, wherein Gaussian noise of a predetermined magnitude is incrementally applied to the original data $x_0$. The goal is to learn a reverse process that gradually removes noise from latent to generate high-quality images. The equations of the posterior and Gaussian transitions are defined as follows:

$$q(x_{1:M}|x_0) := \prod_{t=1}^{M} q(x_t|x_{t-1}), \qquad q(x_t|x_{t-1}) := \mathcal{N}(\sqrt{\frac{\alpha_t}{\alpha_{t-1}}}x_{t-1}, (1 - \frac{\alpha_t}{\alpha_{t-1}})\mathbf{I}) \qquad (1)$$

Here, $[\alpha_0, \ldots, \alpha_M]$ represents a descending sequence of hyperparameters that schedule the noise level, as described in [49]. Through the reparameterization technique [40], we can express $x_t$ as:

$$x_t = \sqrt{\alpha_t}x_0 + \sqrt{1 - \alpha_t}\epsilon, \qquad \epsilon \sim \mathcal{N}(\mathbf{0}, \mathbf{I}) \qquad (2)$$

The aim of a diffusion model is to learn a parameterized distribution $p_\theta(x_0)$ that approximates $q(x_0) = \int p_\theta(x_{0:M})dx_{1:M}$ and is easy to sample from. The parameters $\theta$ are trained by optimizing a variational lower bound to accurately predict the added noise, as outlined in the following objective:

$$\|\epsilon - \epsilon_\theta(\sqrt{\alpha_t}x_0 + \sqrt{1 - \alpha_t}\epsilon, t)\|^2 \qquad (3)$$

While the forward process has $M$ steps, the denoising process can employ a condensed sampling path by choosing a sub-sequence of length $N$ from the set $[1, \ldots, M]$ to expedite the sampling process. DDIM [49] presents a generalized formula to generate a sample $x_{t-1}$ from a sample $x_t$ via:

$$x_{t-1} = \sqrt{\alpha_{t-1}}(\frac{x_t - \sqrt{1 - \alpha_t}\epsilon_\theta(x_t, t)}{\sqrt{\alpha_t}}) + \sqrt{1 - \alpha_{t-1} - \sigma_t^2}\epsilon_\theta(x_t, t) + \sigma_t\epsilon_t, \quad \epsilon_t \sim \mathcal{N}(\mathbf{0}, \mathbf{I}) \quad (4)$$

Varying the choices of parameter $\sigma_t$ yields different denoising processes. Specifically, setting $\sigma_t = \sqrt{\frac{1-\alpha_{t-1}}{1-\alpha_t}}\sqrt{\frac{1-\alpha_t}{\alpha_{t-1}}}$ results in DDPM [23], whereas a $\sigma_t$ of 0 corresponds to DDIM [49].

In previous literature, the notation $t \in [1, ..., N]$ has been conventionally used to index the sampling steps in the denoising process. To distinguish between the original sampling steps and the one that StepbaQ has modified, we employ the notation $[t_1, ..., t_N]$ to represent the original sampling steps, while $[\tau_1, ..., \tau_N]$ is used to denote the steps post-correction.

## 4 Method

In this section, we first present a novel perspective that interprets the distribution shift in the latent space, resulting from quantization errors, as a "stepback" along the sampling trajectory. We then explain how such temporal information affects the sampling trajectory. Finally, we introduce StepbaQ,

a method that calibrates the sampling trajectory and counteracts the negative effects of accumulated quantization error through a sampling step correction mechanism. We demonstrate that StepbaQ can effectively mitigate the accumulation of quantization errors, thereby diminishing the total error. Note that while the derivations discussed herein are based on the denoising process of DDIM [49] as in Eq.4, the proposed StepbaQ is adaptable to other samplers with appropriate modifications.

## 4.1 Quantization Error as Stepback

Following PTQD [20], we assume the quantization error follows Gaussian distribution. Then the conditional probability of the quantized latent variable $\hat{x}_{t_i}$ can be formulated as follows:

$$p(\hat{x}_{t_i}|x_{t_i}) = \mathcal{N}(x_{t_i}, \sigma_i^2), \quad \hat{x}_{t_i} = x_{t_i} + \Delta_i, \quad \Delta_i \sim \mathcal{N}(\mu_i, \sigma_i^2) \tag{5}$$

where $\Delta_i$ denotes the quantization error with $\mu_i$ and $\sigma_i^2$ representing its mean and variance. As the error mean can be readily rectified by subtracting the error mean as described in [20], here we assume $\mu_i = 0$. Recalling the Gaussian transition from step $t_i$ to step $\tau_i$ with $\tau_i \geq t_i$ as mentioned in Eq. 1:

$$q(x_{\tau_i}|x_{t_i}) = \mathcal{N}(\sqrt{\frac{\alpha_{\tau_i}}{\alpha_{t_i}}}x_{t_i}, (1 - \frac{\alpha_{\tau_i}}{\alpha_{t_i}})\mathbf{I}) \tag{6}$$

Scaling the latent variable $x_{\tau_i}$ by $\sqrt{\frac{\alpha_{t_i}}{\alpha_{\tau_i}}}$, we obtain:

$$q(\sqrt{\frac{\alpha_{t_i}}{\alpha_{\tau_i}}}x_{\tau_i}|x_{t_i}) = \mathcal{N}(x_{t_i}, (\frac{\alpha_{t_i}}{\alpha_{\tau_i}} - 1)\mathbf{I}) \tag{7}$$

It is observed that the scaled latent variable $\sqrt{\frac{\alpha_{t_i}}{\alpha_{\tau_i}}}x_{\tau_i}$ shares the same mean as the quantized latent variable $\hat{x}_{t_i}$ as shown in Eq. 5. By selecting a step $\tau_i$ such that $\frac{\alpha_{t_i}}{\alpha_{\tau_i}} - 1 = \sigma_i^2$, or equally $\alpha_{\tau_i} = \frac{\alpha_{t_i}}{1+\sigma_i^2}$, we align the variance of the scaled latent variable with the variance of $\hat{x}_{t_i}$. This alignment allows us to state that the scaled latent variable is statistically equivalent to the quantized latent variable. Consequently, we can consider $\hat{x}_{t_i}$ as being sampled from the distribution at $\tau_i$-th step and scaled by a factor $\sqrt{\frac{\alpha_{t_i}}{\alpha_{\tau_i}}}$. In practice, given that discrete samplers operate with a finite set of sequence $\alpha$, $\tau_i$ can be approximated by solving the following equation (See Appendix B for further discussion):

$$\tau_i = \arg\min_j \|\alpha_j - \frac{\alpha_{t_i}}{1 + \sigma_i^2}\|, \quad j \in [1, \ldots, M] \tag{8}$$

Note that $\tau_i \geq t_i$ due to the descending property of $\alpha$. This novel perspective to treating the quantized latent as if sampled from the distribution at $\tau_i$-th step implies that a temporal shift, or "stepback," occurs in the latent space. This insight enables us to further explore the impact of quantization errors.

## 4.2 Error Arising from Temporal Shift

Based on the finding that a temporal shift occurs in the latent space, we will now demonstrate how such a shift can contribute to error accumulation, resulting in a decline in overall performance.

**Inaccurate Noise Prediction.** Throughout the sampling process, the noise prediction network relies on the temporal feature at the $t_i$-th step as a conditional input, which informs the network about the noise level of the latent variable. Given that the noise level of the quantized latent $\hat{x}_{t_i}$ is increased by quantization error, utilizing the temporal feature at step $t_i$ as a condition could misguide the noise prediction network regarding the actual noise level. This discrepancy has the potential to yield inaccurate noise predictions. An additional concern arises from the distribution shift in the latent space. The noise prediction network is trained with the scheduled latent distributions, as described in Eq. 2. However, quantization error changes the latent variable's mean and variance. Without appropriate adjustment, the distribution of the latent variable $\hat{x}_{t_i}$ deviates from the expected value range. This deviation in the input distribution can also lead to inaccurate noise predictions, ultimately reducing overall performance.

**Trajectory Deviation.** For DDIM, the first term within Eq. 4 indicates the predicted $x_0$. The expression $x_{t_i} - \sqrt{1 - \alpha_{t_i}}\epsilon_\theta(x_{t_i}, t_i)$ can be interpreted as advancing from the latent variable $x_{t_i}$ toward the predicted latent variable $x_0$ along the direction $-\epsilon_\theta(x_{t_i}, t_i)$, with a step size of $\sqrt{1 - \alpha_{t_i}}$.

**Algorithm 1** StepbaQ

---

**Require:** Noise prediction networks $\epsilon_\theta, \hat{\epsilon}_\theta$, sampling steps $[t_1, \ldots, t_N]$, hyperparameters $[\alpha_0, \ldots, \alpha_M]$

1: *# Initialization*
2: $\tau_N = t_N, \hat{x}_{t_N} = x_{t_N}, \quad x_{t_N} \sim \mathcal{N}(\mathbf{0}, \mathbf{I})$
3: **for** $i$ = N,...,2 **do**
4: $\quad \hat{x}_{t_{i-1}} = \sqrt{\alpha_{t_{i-1}}}(\frac{\hat{x}_{t_i} - \sqrt{1-\alpha_{\tau_i}}\hat{\epsilon}_\theta(\hat{x}_{t_i}, \tau_i)}{\sqrt{\alpha_{\tau_i}}}) + \sqrt{1-\alpha_{t_{i-1}}}\hat{\epsilon}_\theta(\hat{x}_{t_i}, \tau_i)$ $\hfill$ *# Quant sampling*
5: $\quad x_{t_{i-1}} = \sqrt{\alpha_{t_{i-1}}}(\frac{x_{t_i} - \sqrt{1-\alpha_{\tau_i}}\epsilon_\theta(x_{t_i}, \tau_i)}{\sqrt{\alpha_{\tau_i}}}) + \sqrt{1-\alpha_{t_{i-1}}}\epsilon_\theta(x_{t_i}, \tau_i)$ $\hfill$ *# Float sampling*
6: $\quad \Delta_{i-1} = \hat{x}_{t_{i-1}} - x_{t_{i-1}}$ $\hfill$ *# Error calculation*
7: $\quad \mu_{i-1}, \sigma_{i-1}^2 = Mean(\Delta_{i-1}), Var(\Delta_{i-1})$
8: $\quad \tau_{i-1} = \arg\min_j \|\alpha_j - \frac{\alpha_{t_{i-1}}}{1+\sigma_{i-1}^2}\|$ $\hfill$ *# Step correction*
9: $\quad$ *# Correct latent if error is large enough*
10: $\quad$ **if** $\tau_{i-1} > t_{i-1}$ **then**
11: $\quad\quad \bar{x}_{t_{i-1}} = \frac{\sqrt{\alpha_{\tau_{i-1}}}}{\sqrt{\alpha_{t_{i-1}}}}(\hat{x}_{t_{i-1}} - \mu_{i-1})$ $\hfill$ *# Latent adjustment*
12: $\quad\quad x_{t_{i-1}} = \hat{x}_{t_{i-1}} = \bar{x}_{t_{i-1}}$ $\hfill$ *# Latent synchronization*
13: $\quad$ **else**
14: $\quad\quad \mu_{i-1} = \mathbf{0}$
15: **return** $[\tau_1, \ldots, \tau_N], [\mu_1, \ldots, \mu_{N-1}]$

---

As the quantized latent variable $\hat{x}_{t_i}$ possesses a higher noise level than $x_{t_i}$, implying that the gap between $x_0$ and $\hat{x}_{t_i}$ is more pronounced, a larger step is required to obtain the expected result. However, previous studies ignore this discrepancy and continue utilizing $\alpha_{t_i}$ during the denoising process. Fig. 1(b) illustrates that adhering to the original step size, without accounting for the quantization error, can lead to a significant accumulation of overall error.

### 4.3 StepbaQ

To tackle the challenges mentioned above, we introduce StepbaQ. By employing the corrected step as obtained by Eq. 8, we demonstrate that StepbaQ can significantly reduce the overall accumulated quantization error through a sequence of simple yet effective modifications.

**Temporal Information Alignment.** The quantized latent variable $\hat{x}_{t_i}$ is characterized by an increased noise level in comparison to its original counterpart $x_{t_i}$. Therefore, the temporal information input must precisely reflect the increased noise levels to aid the noise prediction network make accurate predictions. Replacing the original input $t_i$ with the corrected sampling step $\tau_i$ gives the network a more accurate noise level indication.

**Latent Adjustment.** To approximate the quantized latent $\hat{x}_{t_i}$ as though it were sampled from the distribution at the $\tau_i$-th step, adjustments are necessary to align their distributions. We subtract the channel-wise mean of quantization error from the quantized latent variable, as suggested in PTQD [20], and rescale it by a factor $\sqrt{\frac{\alpha_{\tau_i}}{\alpha_{t_i}}}$. These adjustments effectively bridge the distribution shift and improve the noise prediction results.

**Step Size Adaptation.** Nevertheless, improving the noise prediction results is not enough. As depicted in Fig.2, the quantization errors increase the variance of quantized latent variables and decrease the signal-to-noise ratio (SNR), defined as $SNR = \frac{\mu^2}{\sigma^2}$ [40], which enlarges the distance between $\hat{x}_{t_i}$ and $x_0$. Therefore, the step size should be extended to consider the increased distance. This extension can be realized by substituting the original step size $\sqrt{1-\alpha_{t_i}}$ with $\sqrt{1-\alpha_{\tau_i}}$, which is the requisite step size for progressing towards $x_0$ from the latent variable of $\tau_i$-th step. The adjusted step size is longer than or equal to the original one, owing to the relationship $\alpha_{\tau_i} \leq \alpha_{t_i}$. Fig. 1(c) illustrates that the cumulative error can be markedly diminished by adapting the step size.

**Accounting for Error Accumulation.** Since discrete samplers only involve finite sampling steps, the modifications mentioned above only take effect when quantization errors are substantial enough to result in a corrected step $\tau_i$ larger than $t_i$. No correction will be made if the quantization error does not meet this threshold. Nonetheless, even minor quantization errors can accumulate over multiple steps, imperceptibly altering the latent distribution and degrading the quality of the generated output.

To address this issue, instead of measuring quantization errors at each step independently, StepbaQ measures quantization errors across multiple steps to obtain more accurate statistics for correction.

Algo. 1 shows the overall correction process of StepbaQ. Given floating-point and quantized noise prediction networks $\epsilon_\theta$ and $\hat{\epsilon}_\theta$, StepbaQ calculates the corrected step based on statistics of quantization error. If the error is not substantial enough, no correction is made, allowing the error to accumulate until it becomes sufficiently large for correction. For simplicity, we ignore the iterations over the whole calibration set during the measurement of quantization error. The process yields a sequence of corrected steps $[\tau_1, \ldots, \tau_N]$ and a corresponding sequence of error means $[\mu_1, \ldots, \mu_{N-1}]$, which are subsequently applied during the inference phase. The inference phase is identical to Algo. 1 except that lines 5-8 are skipped. As StepbaQ precludes the accumulation of error, the final output is influenced only by the quantization error present in the last sampling step. This error is significantly less than the original accumulated error without corrections.

# 5 Experiment

We demonstrate the effectiveness of the proposed StepbaQ in enhancing the performance of quantized diffusion models for text-guided image generation. Experiments utilizing an off-the-shelf quantization tool show that StepbaQ can improve the performance of an existing quantized diffusion model without altering its quantization configuration. We also provide comparative results to emphasize StepbaQ's superiority over existing approaches. For qualitative results, please refer to Appendix D.

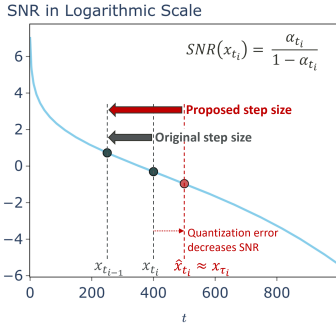

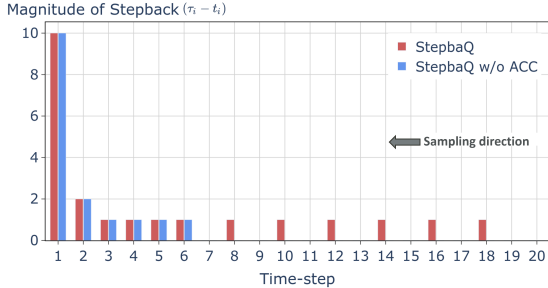

Figure 2: SNR curve of diffusion process. The quantization error decreases the SNR, which could be regarded as a stepback. To address this issue, StepbaQ takes a larger step to reach the scheduled SNR.

Figure 3: Magnitude of stepback for SD v1.5 on the SD-prompts dataset under W8A8 setting. Most sampling step corrections occur at the last few steps of sampling, showing the importance of these steps. Since StepbaQ considers the error accumulation, it performs corrections more frequently than StepbaQ w/o ACC.

**Datasets and Evaluation Metrics.** Our experiments are conducted on two datasets: MS-COCO [36] and Stable-Diffusion-Prompts [7] (SDprompts), each with 5,000 samples for evaluation. We employ the Frechet Inception Distance [22] (FID) as our evaluation metric to assess the quality of images generated by the quantized diffusion models. Notably, as our objective is to minimize the performance discrepancy between the floating-point model and the quantized model, we follow the setting in [50] and utilize the images produced by the floating-point model as the reference for FID evaluation. This approach offers a more precise indication of the performance gap resulting from quantization. In addition, to evaluate the alignment between text and generated images, we also present the CLIP score [21], employing ViT-L/14 [12] as the backbone.

**Implementation Details.** In implementing StepbaQ, we utilize a compact dataset of 128 samples for correction. Empirical observations indicate that this sample size is adequate for gathering the necessary statistics of quantization error, while expanding the sample size to 1024 does not yield a discernible enhancement in performance. The correction can be fast since only a small calibration dataset is required. For example, The correction process for the 20-step Stable-Diffusion v1.5 [9] model can be completed in approximately 20 minutes on a single A6000 GPU.

Table 1: Quantization results of SD v1.5 and SDXL-Turbo on MS-COCO and SDprompts.

| Method | bit (W/A) | SD v1.5 | | | | SDXL-Turbo | | | |
|---|---|---|---|---|---|---|---|---|---|
| | | MS-COCO | | SDprompts | | MS-COCO | | SDprompts | |
| | | FID↓ | CLIP↑ | FID↓ | CLIP↑ | FID↓ | CLIP↑ | FID↓ | CLIP↑ |
| Naive PTQ | 8/8 | 16.69 | 26.93 | 23.66 | 28.16 | 10.48 | 26.93 | 10.12 | 28.05 |
| PTQD | 8/8 | 16.36 | 26.93 | 23.50 | 28.49 | 9.58 | **27.20** | 10.67 | 28.20 |
| StepbaQ | 8/8 | **12.34** | **27.01** | **16.36** | **28.69** | **9.53** | 27.08 | **9.72** | **28.64** |
| Naive PTQ | 4/8 | 95.51 | 23.27 | 134.91 | 19.95 | 44.76 | 26.19 | 45.45 | 28.12 |
| PTQD | 4/8 | 94.55 | 23.47 | 134.62 | 19.77 | 27.77 | 26.70 | 32.38 | 27.98 |
| StepbaQ | 4/8 | **69.43** | **24.12** | **101.31** | **21.71** | **23.92** | **26.74** | **28.14** | **28.46** |

## 5.1 Improvement upon Off-the-Shelf Tool

To facilitate efficient on-device inference of deep learning models, the industry has proposed toolkits such as NeuroPilot [3], OpenVINO [2], SNPE [6], and TensorRT [4]. These tools are tailored to different platforms and support various quantization settings and algorithms. In this section, we utilize the post-training quantization tool in NeuroPilot and demonstrate that StepbaQ significantly improves the performance of the given quantized diffusion model without modifying the original quantization settings. This underscores StepbaQ's effectiveness in seamlessly enhancing model performance within current quantization frameworks. Considering that PTQD depends solely on the statistics of quantization error, similar to our approach, we also include a comparison with PTQD.

**Quantization Settings.** Following previous works, we focus on quantizing the noise prediction network since it is the primary computation workload. Other parts of the diffusion pipelines remain at a floating-point precision. Notation $WxAy$ indicates the weights are quantized to $x$-bit while the activations are quantized to $y$-bit. We test two bit-width settings, W8A8 and W4A8, using the default symmetric/asymmetric settings for weights and activations, respectively. The calibration set comprises 270 samples, which are uniformly selected from each step. NeuroPilot facilitates quantization of most operators within the noise prediction network, e.g., convolution, linear layer, batch matrix multiplication, addition, multiplication, softmax, and SiLU. This scenario presents a more difficult challenge than the one adopted by Q-Diffusion, which only accounts for the quantization of selected operators such as convolution and linear layer. Hence, our setting more rigorously tests StepbaQ's robustness under comprehensive and demanding conditions. Note that for SDXL-Turbo, due to its wide activation value ranges, the input prompt embedding is quantized to 16-bit instead to prevent loss of conceptual information. The consuming linear layers take 16-bit prompt embedding and output 8-bit activations; the weights are still quantized to 4-bit/8-bit as described.

**Stable-Diffusion v1.5 [9].** We perform experiments on Stable-Diffusion v1.5, utilizing a DDIM sampler with 20 sampling steps. We opt for DDIM over the default PNDM[38] sampler as our empirical findings suggest that DDIM yields better results in a 20-step sampling setting. The results are provided in Table 1, which indicate that our proposed correction technique enhances performance in terms of FID and CLIP-score compared to simple post-training quantization. Specifically, for the W8A8 setting, StepbaQ markedly improves FID by 4.35 on the MS-COCO dataset and 7.30 on the SDprompts dataset. Meanwhile, PTQD, constrained by its limited applicability, achieves only modest improvements of 0.33 and 0.16 in FID on the MS-COCO and SDprompts datasets, respectively. For the W4A8 setting, the impact of quantization error is profoundly detrimental, as evidenced by the significant deterioration in both FID and CLIP-score. In this scenario, StepbaQ dramatically reduces the FID by 26.08 and 33.60 on the MS-COCO and SDprompts datasets, respectively. These results underscore the efficacy of StepbaQ, demonstrating its capability to substantially mitigate the adverse effects of quantization error, even under challenging scenarios.

**SDXL-Turbo [5].** To explore whether StepbaQ is compatible with few-step approaches, we conduct experiments on SDXL-Turbo with 4 sampling steps, employing the default EulerAncestralDiscreteScheduler (Euler-a) [1] (See Appendix C for implementation details). Experimental results are presented in Table 1. Interestingly, SDXL-Turbo exhibits greater resilience to quantization errors than SD v1.5. As Euler-a is a stochastic sampler, the error absorption strategy proposed by PTQD is applicable in this setting. The results show that both PTQD and StepbaQ significantly improve the

quality of the generated images under the challenging W4A8 condition, with StepbaQ outperforming PTQD. StepbaQ's improvements, which reduce the FID from 44.76 to 23.92 and from 45.45 to 28.14, effectively eliminate the visual artifacts associated with quantization errors, as shown in Fig. 7.

**Ablation Study.** To assess the contribution of each component of StepbaQ, we perform an ablation study on Stable-Diffusion v1.5 under the W8A8 setting. Table 2 details the results of the ablation study, examining individual components such as Temporal Information Alignment (TIA), Latent Adjustment (LA), Step Size Adaptation (SSA), and Error Accumulation (ACC). The results indicate that mere adjustments to the temporal embedding yield only marginal improvements. In contrast, simultaneous corrections to both temporal embedding and latent variables lead to more substantial enhancements. The result also underscores the critical role of the step size adaptation. Correcting the step size results in significant FID improvements of 2.27 and 2.99 on the MS-COCO and SDprompts datasets, respectively. Finally, accounting for error accumulation enables the model to achieve further FID reductions of 1.20 and 3.22 on the MS-COCO and SDprompts datasets, respectively. This highlights the value of considering error accumulation in improving model performance. Fig. 3 shows the magnitude of stepback for SD v1.5 on the SDprompts dataset under the W8A8 setting. Without considering error accumulation, steps with minor quantization errors are ignored, and therefore, corrections are not applied until the final six sampling steps. In contrast, StepbaQ, accounting for the error accumulation across multiple steps, corrects the sampling steps more frequently. Notably, both configurations exhibit a greater magnitude of stepback during the final stages of the process. This observation implies that the latter sampling steps are particularly susceptible to quantization errors, confirmed by the steep SNR curve as depicted in Fig. 2.

Table 2: Ablation Study of SD v1.5 on MS-COCO and SDprompts under W8A8 setting.

| TIA | LA | SSA | ACC | MS-COCO | | SDprompts | |
|---|---|---|---|---|---|---|---|
| | | | | FID↓ | CLIP↑ | FID↓ | CLIP↑ |
| | | | | 16.69 | 26.93 | 23.66 | 28.16 |
| ✓ | | | | 16.25 | **27.01** | 23.21 | 28.24 |
| ✓ | ✓ | | | 15.81 | 26.87 | 22.57 | 28.53 |
| ✓ | ✓ | ✓ | | 13.54 | 26.82 | 19.58 | 28.61 |
| ✓ | ✓ | ✓ | ✓ | **12.34** | 27.01 | **16.36** | **28.69** |

Table 3: Quantization results of SD v1.4 on MS-COCO and SDprompts.

| Method | bit(W/A) | MS-COCO | | SDprompts | |
|---|---|---|---|---|---|
| | | FID↓ | CLIP↑ | FID↓ | CLIP↑ |
| Q-Diffusion | 8/8 | 8.74 | 26.63 | 9.54 | 29.20 |
| PTQD | 8/8 | 9.07 | 26.58 | 10.22 | 29.18 |
| TFMQ-DM | 8/8 | 8.63 | 26.60 | 9.41 | 29.21 |
| StepbaQ | 8/8 | **8.23** | **26.73** | **8.81** | **29.24** |
| Q-Diffusion | 4/8 | 10.04 | 26.59 | 12.50 | 28.93 |
| PTQD | 4/8 | 10.23 | 26.51 | 11.82 | 28.92 |
| TFMQ-DM | 4/8 | 10.37 | 26.48 | 12.73 | 28.76 |
| StepbaQ | 4/8 | **9.61** | **26.68** | **11.19** | **29.02** |

## 5.2 Comprehensive Comparative Results

To demonstrate the superiority of StepbaQ over existing approaches, we conduct a comparative experiment against Q-diffusion [33], TFMQ-DM [24], and PTQD [20]. We select Q-diffusion as the baseline and integrate StepbaQ and PTQD with it. The experiment is implemented based on the codebase of Q-diffusion [2], adopting the exact quantization setting for fair comparison. It is important to note that we deactivate the timestep-specific quantization parameters utilized in TFMQ-DM. While the technique can potentially enhance the results, it can be regarded as an orthogonal strategy that alters the quantization setting by introducing control of finer granularity.

**Stable-Diffusion v1.4 [8].** Table 3 provides the results of Stable-Diffusion v1.4 using DDIM sampler with 20 sampling steps. Results show that StepbaQ surpasses previous works in terms of FID and CLIP-score across all tested scenarios. Observations indicate that PTQD generally produces results that are inferior to the Q-Diffusion baseline. Further examination revealed that the performance drop is attributable to their strategy of correlated noise elimination, which depends on the correlation between the model's output and the quantization error. However, the observed $R^2$ values are relatively low, indicating a weak correlation. Consequently, the estimated $k$ value utilized for correlated noise elimination is unreliable and may inadvertently compromise the model's performance. On the other hand, TFMQ-DM outperforms Q-Diffusion only under the W8A8 setting through their temporal information aware reconstruction. This limitation stems from their oversight of the distribution shift in the latent space, which is a more critical issue for quantized diffusion models.

# 6 Conclusion

In this work, we introduce a novel aspect that considers the temporal shift in the latent space caused by the quantization error as a "stepback" in the denoising process. We present how this temporal shift would lead to deviation from the scheduled sampling trajectory and harm the performance of quantized diffusion models. To address this issue, we propose StepbaQ, which corrects the sampling steps to calibrate the sampling trajectory and alleviate quantization error accumulation. Our experimental results demonstrate that StepbaQ can serve as a plug-and-play technique, enhancing the performance of a given quantized diffusion model without modifying its quantization settings. The comparative result also demonstrates the superiority of StepbaQ over existing methods. Notably, StepbaQ requires only statistics of quantization error derived from a small calibration dataset and can be applied to both stochastic and deterministic samplers, demonstrating its applicability. However, due to the design of the stepback mechanism, StepbaQ does not apply to single-step scenarios, such as 1-step SDXL-Turbo. How to prevent performance drop of quantized diffusion model under a single-step scenario is an area that requires further exploration in future research.

## Broader Impact.

StepbaQ significantly improves the performance of quantized diffusion models, thereby enabling users to generate high-quality results even on computationally limited edge devices. While this advancement broadens the accessibility of diffusion models to a broader audience, it also increases the risk of these models being leveraged for malicious purposes.

## Acknowledgement

This work was supported by the CAI2 Department at MediaTek, which provided technical support and resources.

## Footnotes

[2]https://github.com/Xiuyu-Li/q-diffusion

## References

[1] EulerAncestralDiscreteScheduler. https://huggingface.co/docs/diffusers/api/schedulers/euler_ancestral.

[2] Intel OpenVINO. https://software.intel.com/content/www/us/en/develop/tools/openvino-toolkit.html.

[3] MediaTek NeuroPilot. https://neuropilot.mediatek.com.

[4] NVIDIA TensorRT. https://developer.nvidia.com/tensorrt.

[5] SDXL-Turbo. https://huggingface.co/stabilityai/sdxl-turbo.

[6] Snapdragon Neural Processing Engine SDK. https://developer.qualcomm.com/sites/default/files/docs/snpe/index.html.

[7] Stable-diffusion prompts dataset. https://huggingface.co/datasets/Gustavosta/Stable-Diffusion-Prompts.

[8] Stable-Diffusion v1.4. https://huggingface.co/CompVis/stable-diffusion-v1-4.

[9] Stable-Diffusion v1.5. https://huggingface.co/runwayml/stable-diffusion-v1-5.

[10] Y. Cai, Z. Yao, Z. Dong, A. Gholami, M. W. Mahoney, and K. Keutzer. Zeroq: A novel zero shot quantization framework. In *Proceedings of the IEEE/CVF Conference on Computer Vision and Pattern Recognition*, pages 13169–13178, 2020.

[11] P. Dhariwal and A. Nichol. Diffusion models beat gans on image synthesis. *Advances in neural information processing systems*, 34:8780–8794, 2021.

[12] A. Dosovitskiy, L. Beyer, A. Kolesnikov, D. Weissenborn, X. Zhai, T. Unterthiner, M. Dehghani, M. Minderer, G. Heigold, S. Gelly, et al. An image is worth 16x16 words: Transformers for image recognition at scale. *International Conference on Learning Representations*, 2021.

[13] S. K. Esser, J. L. McKinstry, D. Bablani, R. Appuswamy, and D. S. Modha. Learned step size quantization. *International Conference on Learning Representations*, 2020.

[14] G. Fang, X. Ma, and X. Wang. Structural pruning for diffusion models. *Advances in neural information processing systems*, 36, 2024.

[15] E. Frantar, S. Ashkboos, T. Hoefler, and D. Alistarh. Gptq: Accurate post-training quantization for generative pre-trained transformers. *arXiv preprint arXiv:2210.17323*, 2022.

[16] A. Gholami, S. Kim, Z. Dong, Z. Yao, M. W. Mahoney, and K. Keutzer. A survey of quantization methods for efficient neural network inference. In *Low-Power Computer Vision*, pages 291–326. Chapman and Hall/CRC, 2022.

[17] R. Gong, X. Liu, S. Jiang, T. Li, P. Hu, J. Lin, F. Yu, and J. Yan. Differentiable soft quantization: Bridging full-precision and low-bit neural networks. In *Proceedings of the IEEE/CVF international conference on computer vision*, pages 4852–4861, 2019.

[18] I. Goodfellow, J. Pouget-Abadie, M. Mirza, B. Xu, D. Warde-Farley, S. Ozair, A. Courville, and Y. Bengio. Generative adversarial nets. *Advances in neural information processing systems*, 27, 2014.

[19] Y. He, J. Liu, W. Wu, H. Zhou, and B. Zhuang. Efficientdm: Efficient quantization-aware fine-tuning of low-bit diffusion models. *International Conference on Learning Representations*, 2024.

[20] Y. He, L. Liu, J. Liu, W. Wu, H. Zhou, and B. Zhuang. Ptqd: Accurate post-training quantization for diffusion models. *Advances in Neural Information Processing Systems*, 36, 2024.

[21] J. Hessel, A. Holtzman, M. Forbes, R. L. Bras, and Y. Choi. Clipscore: A reference-free evaluation metric for image captioning. *Conference on Empirical Methods in Natural Language Processing*, pages 7514–7528, 2021.

[22] M. Heusel, H. Ramsauer, T. Unterthiner, B. Nessler, and S. Hochreiter. Gans trained by a two time-scale update rule converge to a local nash equilibrium. *Advances in neural information processing systems*, 30, 2017.

[23] J. Ho, A. Jain, and P. Abbeel. Denoising diffusion probabilistic models. *Advances in neural information processing systems*, 33:6840–6851, 2020.

[24] Y. Huang, R. Gong, J. Liu, T. Chen, and X. Liu. Tfmq-dm: Temporal feature maintenance quantization for diffusion models. *Proceedings of the IEEE/CVF Conference on Computer Vision and Pattern Recognition*, 2024.

[25] I. Hubara, Y. Nahshan, Y. Hanani, R. Banner, and D. Soudry. Accurate post training quantization with small calibration sets. In *International Conference on Machine Learning*, pages 4466–4475. PMLR, 2021.

[26] B. Jacob, S. Kligys, B. Chen, M. Zhu, M. Tang, A. Howard, H. Adam, and D. Kalenichenko. Quantization and training of neural networks for efficient integer-arithmetic-only inference. In *Proceedings of the IEEE conference on computer vision and pattern recognition*, pages 2704–2713, 2018.

[27] S. Jain, A. Gural, M. Wu, and C. Dick. Trained quantization thresholds for accurate and efficient fixed-point inference of deep neural networks. *Proceedings of Machine Learning and Systems*, 2:112–128, 2020.

[28] Z. Jiang, C. Mao, Y. Pan, Z. Han, and J. Zhang. Scedit: Efficient and controllable image diffusion generation via skip connection editing. *Proceedings of the IEEE/CVF conference on computer vision and pattern Recognition*, 2024.

[29] T. Karras, M. Aittala, T. Aila, and S. Laine. Elucidating the design space of diffusion-based generative models. *Advances in Neural Information Processing Systems*, 35:26565–26577, 2022.

[30] D. P. Kingma and M. Welling. Auto-encoding variational bayes. *International Conference on Learning Representations*, 2014.

[31] R. Krishnamoorthi. Quantizing deep convolutional networks for efficient inference: A whitepaper. *arXiv preprint arXiv:1806.08342*, 2018.

[32] B. Li, K. Xue, B. Liu, and Y.-K. Lai. Bbdm: Image-to-image translation with brownian bridge diffusion models. In *Proceedings of the IEEE/CVF conference on computer vision and pattern Recognition*, pages 1952–1961, 2023.

[33] X. Li, Y. Liu, L. Lian, H. Yang, Z. Dong, D. Kang, S. Zhang, and K. Keutzer. Q-diffusion: Quantizing diffusion models. In *Proceedings of the IEEE/CVF International Conference on Computer Vision*, pages 17535–17545, 2023.

[34] Y. Li, R. Gong, X. Tan, Y. Yang, P. Hu, Q. Zhang, F. Yu, W. Wang, and S. Gu. Brecq: Pushing the limit of post-training quantization by block reconstruction. *International Conference on Learning Representations*, 2021.

[35] Y. Li, S. Xu, X. Cao, X. Sun, and B. Zhang. Q-dm: An efficient low-bit quantized diffusion model. *Advances in Neural Information Processing Systems*, 36, 2024.

[36] T.-Y. Lin, M. Maire, S. Belongie, J. Hays, P. Perona, D. Ramanan, P. Dollár, and C. L. Zitnick. Microsoft coco: Common objects in context. In *Computer Vision–ECCV 2014: 13th European Conference, Zurich, Switzerland, September 6-12, 2014, Proceedings, Part V 13*, pages 740–755. Springer, 2014.

[37] X. Lin, J. He, Z. Chen, Z. Lyu, B. Fei, B. Dai, W. Ouyang, Y. Qiao, and C. Dong. Diffbir: Towards blind image restoration with generative diffusion prior. *arXiv preprint arXiv:2308.15070*, 2023.

[38] L. Liu, Y. Ren, Z. Lin, and Z. Zhao. Pseudo numerical methods for diffusion models on manifolds. *International Conference on Learning Representations*, 2022.

[39] C. Lu, Y. Zhou, F. Bao, J. Chen, C. Li, and J. Zhu. Dpm-solver: A fast ode solver for diffusion probabilistic model sampling in around 10 steps. *Advances in Neural Information Processing Systems*, 35:5775–5787, 2022.

[40] C. Luo. Understanding diffusion models: A unified perspective. *arXiv preprint arXiv:2208.11970*, 2022.

[41] S. Luo, Y. Tan, L. Huang, J. Li, and H. Zhao. Latent consistency models: Synthesizing high-resolution images with few-step inference. *arXiv preprint arXiv:2310.04378*, 2023.

[42] M. Nagel, R. A. Amjad, M. Van Baalen, C. Louizos, and T. Blankevoort. Up or down? adaptive rounding for post-training quantization. In *International Conference on Machine Learning*, pages 7197–7206. PMLR, 2020.

[43] Y. Nahshan, B. Chmiel, C. Baskin, E. Zheltonozhskii, R. Banner, A. M. Bronstein, and A. Mendelson. Loss aware post-training quantization. *Machine Learning*, 110(11):3245–3262, 2021.

[44] A. Q. Nichol and P. Dhariwal. Improved denoising diffusion probabilistic models. In *International conference on machine learning*, pages 8162–8171. PMLR, 2021.

[45] R. Rombach, A. Blattmann, D. Lorenz, P. Esser, and B. Ommer. High-resolution image synthesis with latent diffusion models. In *Proceedings of the IEEE/CVF conference on computer vision and pattern recognition*, pages 10684–10695, 2022.

[46] A. Sauer, D. Lorenz, A. Blattmann, and R. Rombach. Adversarial diffusion distillation. *arXiv preprint arXiv:2311.17042*, 2023.

[47] Y. Shang, Z. Yuan, B. Xie, B. Wu, and Y. Yan. Post-training quantization on diffusion models. In *Proceedings of the IEEE/CVF Conference on Computer Vision and Pattern Recognition*, pages 1972–1981, 2023.

[48] J. So, J. Lee, D. Ahn, H. Kim, and E. Park. Temporal dynamic quantization for diffusion models. *Advances in Neural Information Processing Systems*, 36, 2024.

[49] J. Song, C. Meng, and S. Ermon. Denoising diffusion implicit models. *International Conference on Learning Representations*, 2021.

[50] S. Tang, X. Wang, H. Chen, C. Guan, Z. Wu, Y. Tang, and W. Zhu. Post-training quantization with progressive calibration and activation relaxing for text-to-image diffusion models. *arXiv preprint arXiv:2311.06322*, 2023.

[51] J. W. Tukey et al. *Exploratory data analysis*, volume 2. Springer, 1977.

[52] C. Wang, Z. Wang, X. Xu, Y. Tang, J. Zhou, and J. Lu. Towards accurate post-training quantization for diffusion models. *Proceedings of the IEEE/CVF Conference on Computer Vision and Pattern Recognition*, 2024.

[53] J. Wang, Z. Yue, S. Zhou, K. C. Chan, and C. C. Loy. Exploiting diffusion prior for real-world image super-resolution. *arXiv preprint arXiv:2305.07015*, 2023.

[54] B. Xia, Y. Zhang, S. Wang, Y. Wang, X. Wu, Y. Tian, W. Yang, and L. Van Gool. Diffir: Efficient diffusion model for image restoration. In *Proceedings of the IEEE/CVF International Conference on Computer Vision*, pages 13095–13105, 2023.

[55] G. Xiao, J. Lin, M. Seznec, H. Wu, J. Demouth, and S. Han. Smoothquant: Accurate and efficient post-training quantization for large language models. In *International Conference on Machine Learning*, pages 38087–38099. PMLR, 2023.

[56] S. Xie, Z. Zhang, Z. Lin, T. Hinz, and K. Zhang. Smartbrush: Text and shape guided object inpainting with diffusion model. In *Proceedings of the IEEE/CVF Conference on Computer Vision and Pattern Recognition*, pages 22428–22437, 2023.

[57] Y. Yang, X. Dai, J. Wang, P. Zhang, and H. Zhang. Efficient quantization strategies for latent diffusion models. *arXiv preprint arXiv:2312.05431*, 2023.

[58] Z. Yue, J. Wang, and C. C. Loy. Resshift: Efficient diffusion model for image super-resolution by residual shifting. *Advances in Neural Information Processing Systems*, 36, 2024.

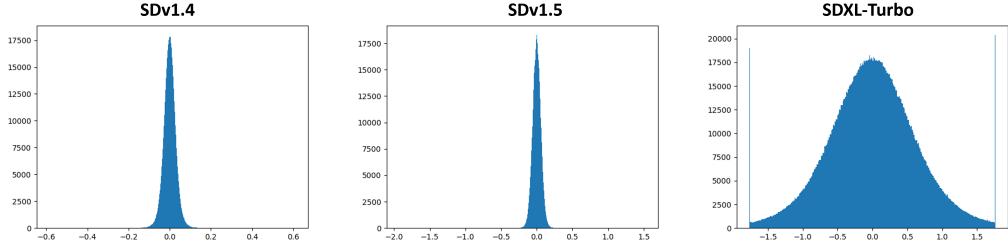

Figure 4: The distribution of quantization errors collected under W8A8 setting.

# A  Analysis of Quantization Error

Our assumption that quantization error follows a Gaussian distribution is inspired by the empirical findings presented in PTQD [20], where the authors verify the Gaussian nature of quantization error through statistical tests detailed in Appendix B of their paper. In our own empirical investigations, we observed that the distribution of quantization error typically exhibits a symmetric, bell-shaped curve, as depicted in Fig.4. This observation aligns with the result shown in Fig.3 of the PTQD paper.

To delve deeper into the characteristics of quantization error, we visualize the error distribution in Fig.4 and present an analysis of both the kurtosis (Fisher) and skewness of the error distribution in Table 4, averaged across all steps for each model under the W8A8 setting. Our findings indicate that the skewness values are notably small, confirming our observation that the quantization error follows a symmetric, bell-shaped distribution. Regarding kurtosis, the values observed are greater than zero for each model, suggesting that the distribution of quantization error exhibits relatively fat tails. These fat tails likely originate from clipping errors, which occur less frequently but are more significant than rounding errors.

Although the quantization error exhibits higher kurtosis than a Gaussian distribution, our proposed StepbaQ method still significantly enhances the performance of diffusion models, as evidenced by the results in Tables 1 and 3 of our main paper. This underscores the robustness and applicability of our approach, demonstrating that StepbaQ can effectively improve model performance under realistic conditions.

Table 4: The kurtosis and skewness of the quantization errors collected under W8A8 setting.

|  | SDv1.4 | SDv1.5 | SDXL-Turbo |
|---|---|---|---|
| Kurtosis | 2.627 | 3.429 | 0.502 |
| Skewness | 0.036 | -0.020 | -0.007 |

# B  Discussion of Quantization Error as Stepback

Here, we provide another derivation of Eq. 8 from the signal-to-noise ratio (SNR) perspective. Combining Eq. 2 and Eq. 5, the quantized latent variable $\hat{x}_{t_i}$ can be expressed as:

$$\hat{x}_{t_i} = x_{t_i} + \Delta_i = \sqrt{\alpha_{t_i}}x_0 + \sqrt{1 - \alpha_{t_i}}\epsilon + \Delta_i \tag{9}$$

Given that the sum of two independent Gaussian random variables remains a Gaussian with mean being the sum of the individual means and variance being the sum of the individual variances, the quantized latent $\hat{x}$ can be represented as:

$$\hat{x}_{t_i} = \sqrt{\alpha_{t_i}}x_0 + \sqrt{1 - \alpha_{t_i} + \sigma_i^2}\epsilon', \qquad \epsilon' \sim \mathcal{N}(\mathbf{0}, \mathbf{I}) \tag{10}$$

From Eq. 5, it is evident that the quantization error amplifies the variance of the quantized latent variable $\hat{x}_{t_i}$. This increase in variance, in turn, diminishes the SNR of the latent variable, which is defined as $SNR = \frac{\mu^2}{\sigma^2}$ [40], resulting in:

$$SNR(x_{t_i}) = \frac{\alpha_{t_i}}{1 - \alpha_{t_i}}, \quad SNR(\hat{x}_{t_i}) = \frac{\alpha_{t_i}}{1 - \alpha_{t_i} + \sigma_i^2} \tag{11}$$

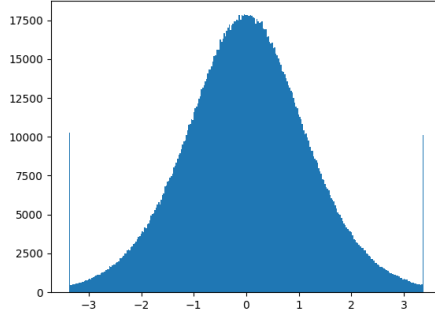

Figure 5: The distribution of quantization error collected from W4A8 SDXL-Turbo on SDprompts dataset, outliers are clipped by Tukey's fence with $k$ set as 1.7.

Since diffusion models exhibit an increasing SNR in the denoising process, this reduction in SNR can be interpreted as a "stepback," as depicted in Fig. 2. Therefore, we can find a step $\tau_i$ that better aligns with the SNR of $\hat{x}_{t_i}$, such that:

$$\tau_i = \arg\min_j \|SNR(x_j) - SNR(\hat{x}_{t_i})\|, \quad j \in [1, \ldots, M] \tag{12}$$

This allows us to approximate $\hat{x}_{t_i}$ as if sampled from the distribution at the $\tau_i$-th step. By reformulating Eq. 12 as follows:

$$
\begin{aligned}
\tau_i &= \arg\min_j \left\| \frac{\alpha_j}{1 - \alpha_j} - \frac{\alpha_{t_i}}{1 - \alpha_{t_i} + \sigma_i^2} \right\| \\
&= \arg\min_j \left\| \alpha_j - \frac{\alpha_{t_i}}{1 + \sigma_i^2} \right\|, \quad j \in [1, \ldots, M]
\end{aligned}
\tag{13}
$$

we arrive at the same objective described in Eq. 8.

## C  Extending StepbaQ to SDXL-Turbo

We have demonstrated the application of StepbaQ to diffusion models using DDIM sampler in Sec. 4. Here, we introduce several modifications to adapt StepbaQ for SDXL-Turbo, which employs the EulerAncestralDiscreteScheduler (Euler-a) as its sampler.

**Statistical Analysis of Quantization Error.** In our analysis of SDXL-Turbo's quantization error, we find outliers with large values, making the distribution of quantization error very long-tailed. To mitigate the influence of these outliers on the variance measurement, we employ Tukey's fence [51], empirically setting the value of $k$ to 1.7, which effectively clips out these extreme values, as shown in Fig. 5.

**Finding the Corrected Step.** The Euler-a scheduler determines the noise level at each step using the equation $\sigma(t_i) = \sqrt{\frac{1 - \alpha_{t_i}}{\alpha_{t_i}}}$. To identify the corrected sampling step $\tau_i$ that more accurately represents the noise level of the quantized latent variable $\hat{x}_{t_i}$, we solve the following equation:

$$\tau_i = \arg\min_j \|\sigma(j)^2 - (\sigma(t_i)^2 + \sigma_i^2)\|, \quad j \in [1, \ldots, M] \tag{14}$$

where $\sigma_i^2$ represents the variance of the quantization error.

**Temporal Information Alignment.** As discussed in Sec. 4, the time embedding input provides crucial information about the noise level to the noise prediction network. Aligning this temporal information with the latent variable is generally beneficial for improving noise prediction. However, for SDXL-Turbo, aligning temporal information does not enhance model prediction. We hypothesize that this is because the training procedure of SDXL-Turbo [46] involves only four selected steps. In this case, correcting the temporal input would introduce inputs unseen during training. Therefore, we deactivate the temporal information alignment component when integrating StepbaQ with SDXL-Turbo.

**Latent Adjustment** With Euler-a as its sampler, SDXL-Turbo rescales the latent variable $x$ before feeding it into the noise prediction network by a factor $s(t_i)$ (Eq.170 in [29]). To account for the increased noise level in the quantized latent variable $\hat{x}_{t_i}$, we use the scaling factor of the corrected step $s(\tau_i)$, which more accurately reflects the noise level.

**Step Size Correction** The original step size defined by Euler-a for the $t_i$-th step moving toward the target step with noise level $\sigma_{down}$ is defined as $\sigma(t_i) - \sigma_{down}$. To accommodate the increased noise level of the quantized latent variable $\hat{x}_{t_i}$, we employ a larger step size, $\sigma(\tau_i) - \sigma_{down}$. This adjustment ensures that the resulting noise level aligns with the intended target.

## D    Qualitative Results

In this section, we present the qualitative results of our experiments. Figure 6 displays the outcomes for SD v1.5, which is quantized using NeuroPilot on the SDprompts dataset under the W8A8 setting. Compared to the results from Naive PTQ and PTQD, our proposed StepbaQ method effectively addresses the inconsistent visual artifacts introduced by quantization errors while preserving the overall structural integrity of the images.

Figure 7 illustrates the results for SDXL-Turbo, also quantized by NeuroPilot on SDprompts but under the more challenging W4A8 setting. In this scenario, images processed with Naive PTQ display noticeable crack-like artifacts. Both PTQD and StepbaQ manage to smooth these visual disturbances. However, the images produced by StepbaQ demonstrate a higher degree of visual smoothness and greater similarity to those generated by the floating-point model, underscoring the effectiveness of StepbaQ.

Figure 8 shows the comparative results for SD v1.4 on SDprompts under the W4A8 setting. This experiment was conducted using the Q-Diffusion codebase, which represents a less challenging quantization scenario as described in Sec. 5.1. Despite the less pronounced differences in image quality, StepbaQ consistently delivers results that more closely resemble those of the floating-point model, avoiding the artifacts that are sometimes evident in outputs from other methods.

FP        PTQ        PTQD        StepbaQ

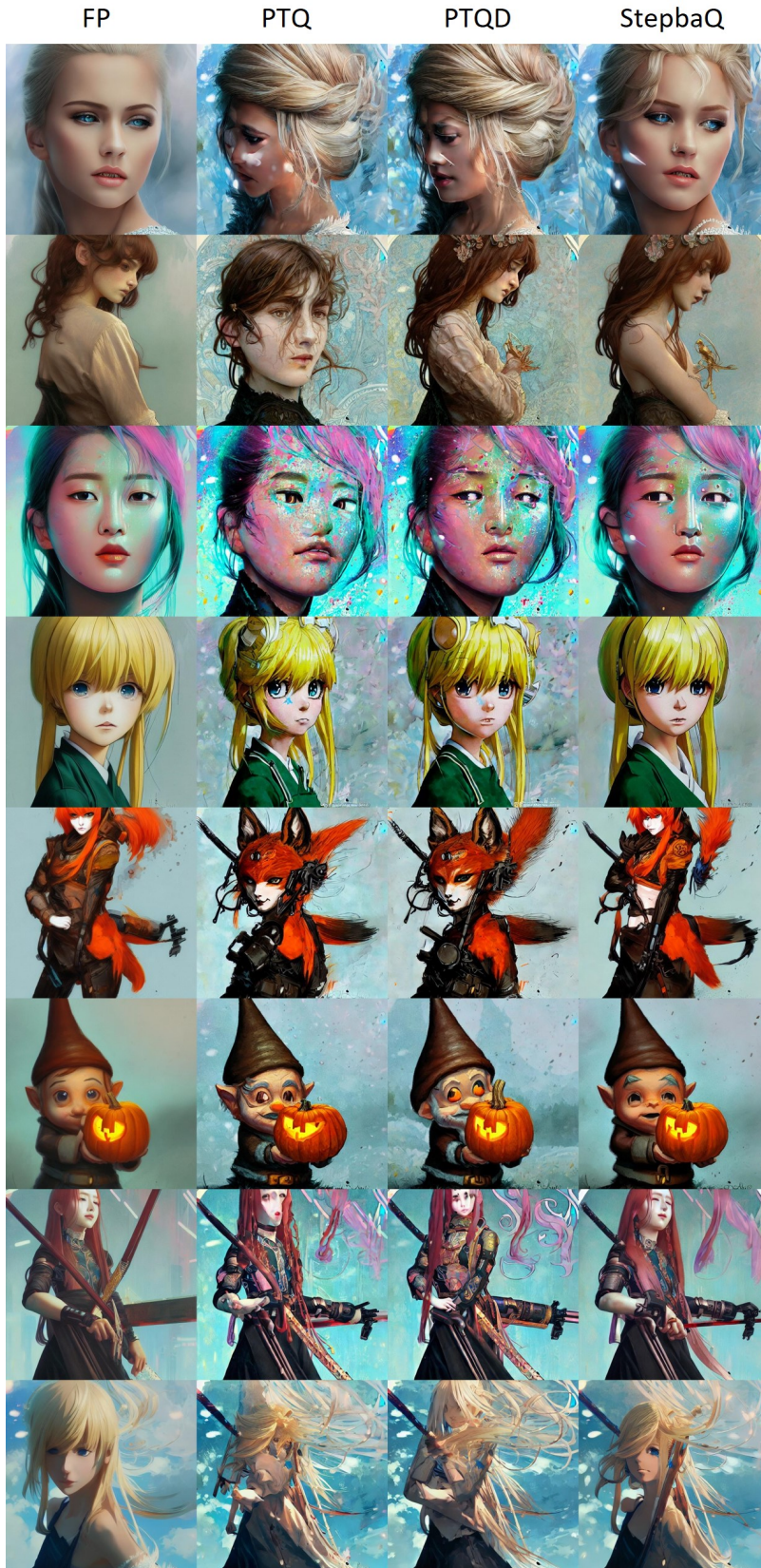

Figure 6: Qualitative results of SD v1.5 on SDprompts under W8A8 setting.

FP          PTQ          PTQD          StepbaQ

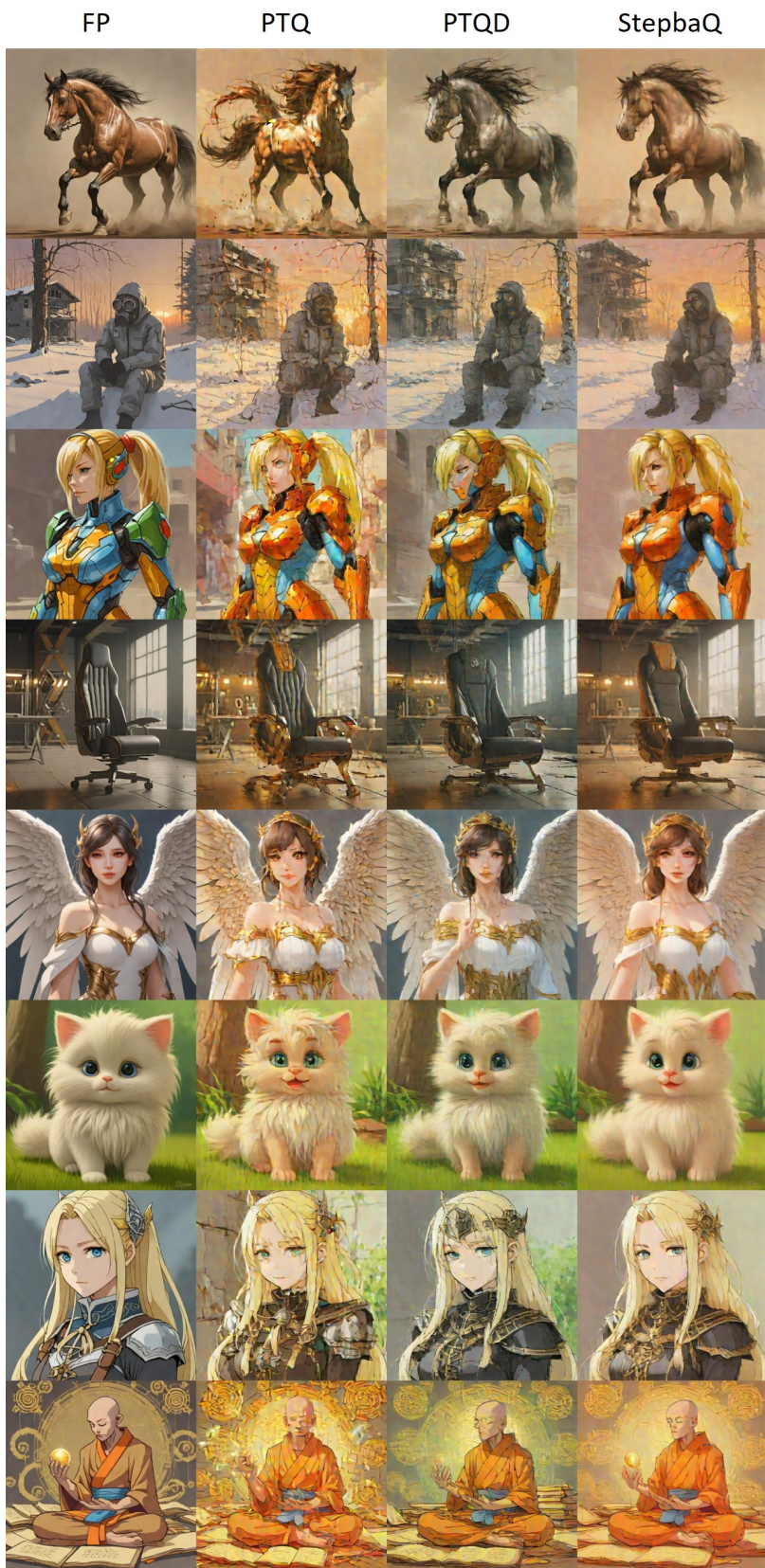

Figure 7: Qualitative results of SDXL-Turbo on SDprompts under W4A8 setting.

FP          Q-Diffusion          PTQD          TFMQ          StepbaQ

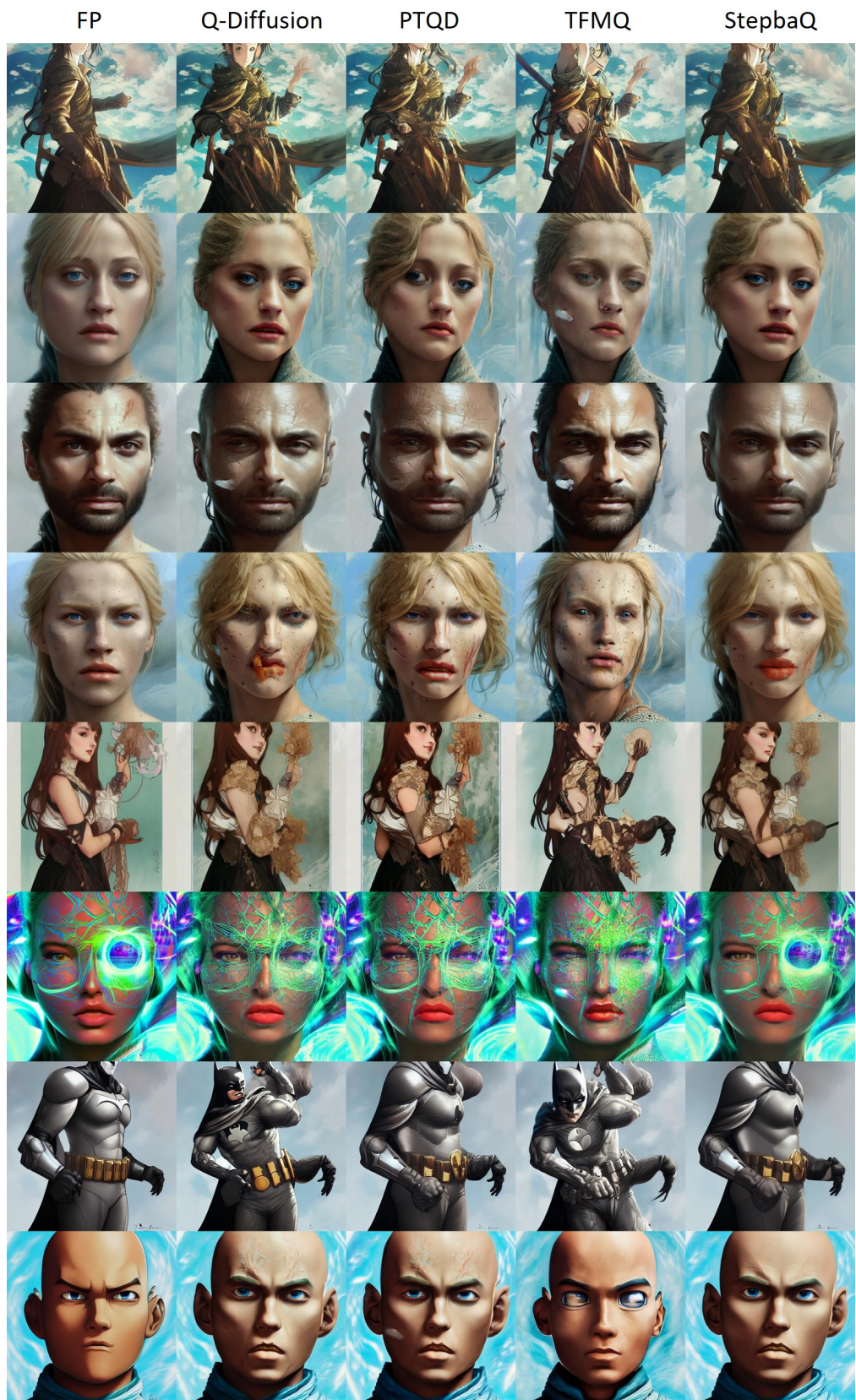

Figure 8: Qualitative results of SD v1.4 on SDprompts under W4A8 setting.

